# Kernels on Attributed Pointsets with Applications

**Mehul Parsana**[1]
mehul.parsana@gmail.com

**Sourangshu Bhattacharya**[1]
sourangshu@gmail.com

**Chiranjib Bhattacharyya**[1]
chiru@csa.iisc.ernet.in

**K. R. Ramakrishnan**[2]
krr@ee.iisc.ernet.in

## Abstract

This paper introduces kernels on attributed pointsets, which are sets of vectors embedded in an euclidean space. The embedding gives the notion of neighborhood, which is used to define positive semidefinite kernels on pointsets. Two novel kernels on neighborhoods are proposed, one evaluating the attribute similarity and the other evaluating shape similarity. Shape similarity function is motivated from spectral graph matching techniques. The kernels are tested on three real life applications: face recognition, photo album tagging, and shot annotation in video sequences, with encouraging results.

## 1 Introduction

In recent times, one of the major challenges in kernel methods has been design of kernels on structured data e.g. sets [9, 17, 15], graphs [8, 3], strings, automata, etc. In this paper, we propose kernels on a type of structured objects called *attributed pointsets* [18]. Attributed pointsets are points embedded in a euclidean space with a vector of attributes attached to each point. The embedding of points in the euclidean space yields a notion of *neighborhood* of each point which is exploited in designing new kernels. Also, we describe the notion of similarity between pointsets which model many real life scenarios and incorporate it in the proposed kernels.

The main contribution of this paper is definition of two different kernels on neighborhoods. These neighborhood kernels are then used to define kernels on the entire pointsets. The first kernel treats the neighborhoods as sets of vectors for calculating the similarity. Second kernel calculates similarity in shape of the two neighborhoods. It is motivated using spectral graph matching techniques [16].

We demonstrate practical applications of the kernels on the well known task of face recognition [20], and two other novel tasks of tagging photo albums and annotation of shots in video sequences. For the face recognition task, we test our kernels on benchmark datasets and compare their performance with state-of-the-art algorithms. Our kernels outperform the existing methods in many cases. The kernels also perform according to expectation on the two novel applications. Section 2 defines attributed pointsets and contrasts it with related notions. Section 3 proposes two kernels and section 4 describes experimental results.

## 2 Definition and related work

An attributed pointset [18, 1] (a.k.a. point pattern) $X$ is sets of points in $\mathbb{R}^k$ with attributes or labels (real vectors in this case) attached to each point. Thus, $X = \{(\mathbf{x}_i, \mathbf{d}_i) | i = 1 \ldots n\}$, where $\mathbf{x}_i \in \mathbb{R}^u$ and $\mathbf{d}_i \in \mathbb{R}^v$, $l$ being the dimension of the attribute vector. The number of points in a pointset,

$n$, is variable. Also, for practical purposes pointsets with $u = 2, 3$ are of interest. The construct of pointsets are richer than sets of vectors [17] because of the structure formed by embedding of the points in a euclidean space. However, they are less general than attributed graphs because all attributed graphs cannot be embedded onto a euclidean space. Pointsets are useful in several domains including computer vision [18], computational biology [5], etc.

The notion of similarity between pointsets is also different from those between sets of vectors, or graphs. The main aspect of similarity is that there should be correspondences (1-1 mappings) between the points of a pointset such that the relative positions of corresponding point are same. Also the attribute vectors of the matching points should be similar. In case of sets of vectors, the kernel function captures the similarity between aggregate properties of the two sets, such as the principle angles between spanned subspaces [17], or distance between the distributions generating the vectors [9]. Kernels on graphs try to capture similarity in the graph topology by comparing the number of similar paths [3], or comparing steady state distributions on of linear systems on graphs [8].

For example, consider recognizing faces using local descriptors calculated at some descriptor points (corner points in this case) on the face. It is necessary that subsets of descriptor points found in two images of the same face should be approximately superimposable (slight changes may be due to change of expression) and that the descriptor values for the corresponding points should be roughly same to ensure similar local features. Thus, a face can be modeled as an attributed pointset $X = \{(\mathbf{x}_i, \mathbf{d}_i) | i = 1 \ldots n\}$, where $\mathbf{x}_i \in \mathbb{R}^2$ is the coordinate of $i^{th}$ descriptor point and $\mathbf{d}_i \in \mathbb{R}^v$ is the local descriptor vector at the $i^{th}$ descriptor point. Similar arguments can be provided for any object recognition task.

A local descriptor based kernel was proposed for object recognition in similar setting in [12]. Suppose $X^A = \{(\mathbf{x}_i^A, \mathbf{d}_i^A) | i = 1 \ldots n_A\}$ and $X^B = \{(\mathbf{x}_i^B, \mathbf{d}_i^B) | i = 1 \ldots n_B\}$ are two pointsets. The normalized sum kernel [12] was defined as $\mathcal{K}_{NS}(X^A, X^B) = \frac{1}{n_A n_B} \sum_{i=1}^{n_A} \sum_{j=1}^{n_B} (\mathcal{K}(\mathbf{d}_i^A, \mathbf{d}_j^B))^p$, where $\mathcal{K}(\mathbf{d}_i^A, \mathbf{d}_j^B)$ is some kernel function on the descriptors. It was argued in [12] that raising the kernel to a high power $p$ approximately calculates similarity between matched pairs of vectors. Using the RBF kernel $\mathcal{K}_{RBF}(x, y) = e^{-\frac{\|x-y\|^2}{\sigma^2}}$, and adjusting the parameter $p$ in $\sigma$, we get the normalized sum kernels as:

$$\mathcal{K}_{NS}(X^A, X^B) = \frac{1}{n_A n_B} \sum_{i=1}^{n_A} \sum_{j=1}^{n_B} \mathcal{K}_{RBF}(\mathbf{d}_i^A, \mathbf{d}_j^B) \tag{1}$$

Observe that this kernel doesn't use the in formation in $\mathbf{x}_i$ anywhere, and thus is actually a kernel on a set of vectors. In fact, this kernel can be derived as a special case of the set kernel proposed in [15]. The kernel $\mathcal{K}(A, B) = trace\left(\sum_r (A^T \hat{G}_r B) \hat{F}_r\right)$ becomes $\mathcal{K}(A, B) = \sum_{ij} k(\mathbf{a}_i, \mathbf{b}_j) f_{ij}$ for $\hat{G}_r = I$ and $F = \sum_r F_r$ (whose entries are $f_{ij}$) should be positive semidefinite [15]. Thus, choosing $F = \mathbf{1}\mathbf{1}^T$ (all entries 1) and multiplying the kernel by $\frac{1}{n_A^2 n_B^2}$ and using $\mathcal{K}_{RBF}$ as the kernel on vectors, we get back the kernel defined in (1). The normalized sum kernel is used as the basic kernel for development and validation of the new kernels proposed here. In the next section, we incorporate position $\mathbf{x}_i$ of the points using the concept of neighborhood.

## 3 Kernels

### 3.1 Neighborhood kernels

The key idea in this section is to use spatially co-occurring points of a point to improve the similarity values given by the kernel function. In other words, we hypothesize that similar points from two pointsets should also have neighboring points which are similar. Thus, for each point we define a *neighborhood* of the point and weight the similarity between each pair of points with the similarity between their neighborhoods.

The $k$-neighborhood $\mathcal{N}_i$ of a point $(\mathbf{x}_i, \mathbf{d}_i)$ in a pointset $X$ is defined as the set of points (including itself) that are closest to it in the embedding euclidean space. So, $\mathcal{N}_i = \{(\mathbf{x}_j, \mathbf{d}_j) \in X | \|\mathbf{x}_i - \mathbf{x}_j\| \leq \|\mathbf{x}_i - \mathbf{x}_l\| \forall (\mathbf{x}_l, \mathbf{d}_l) \notin \mathcal{N}_i$ and $|\mathcal{N}_i| = k\}$. The neighborhood kernel between two points $(\mathbf{x}_i^A, \mathbf{d}_i^A)$

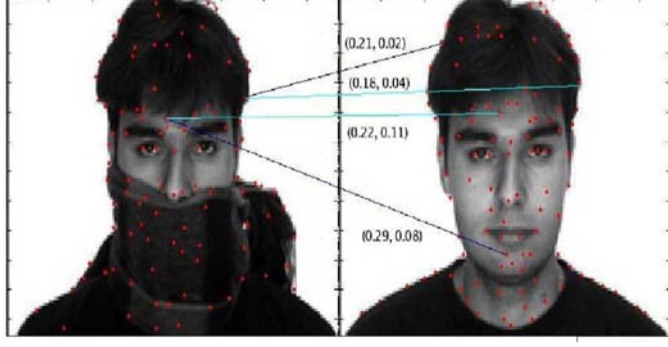

Figure 1: Correspondences implicitly found by sum and neighborhood kernels

and $(\mathbf{x}_j^B, \mathbf{d}_j^B)$ is defined as:

$$\mathcal{K}_N((\mathbf{x}_i^A, \mathbf{d}_i^A), (\mathbf{x}_j^B, \mathbf{d}_j^B)) = \mathcal{K}_{RBF}(\mathbf{d}_i^A, \mathbf{d}_j^B) \times \frac{1}{|\mathcal{N}_i^A||\mathcal{N}_j^B|} \sum_{(\mathbf{x}_s^A, \mathbf{d}_s^A) \in \mathcal{N}_i^A} \sum_{(\mathbf{x}_t^B, \mathbf{d}_t^B) \in \mathcal{N}_j^B} \mathcal{K}_{RBF}(\mathbf{d}_s^A, \mathbf{d}_t^B)$$

(2)

The neighborhood kernel (NK) between two pointsets $X^A$ and $X^B$ is thus defined as:

$$\mathcal{K}_{NK}(X^A, X^B) = \frac{1}{n_A n_B} \times \sum_{i=1}^{n_A} \sum_{j=1}^{n_B} \mathcal{K}_N((\mathbf{x}_i^A, \mathbf{d}_i^A), (\mathbf{x}_j^B, \mathbf{d}_j^B))$$

(3)

It is easy to see that $\mathcal{K}_{NK}$ is a positive semidefinite kernel function. Even though $\mathcal{K}_{NK}$ is a straightforward extension, it considerably improves accuracy of $\mathcal{K}_{NS}$. Figure 1 shows values of $\mathcal{K}_{NS}$ and $\mathcal{K}_{NK}$ for 4 pairs of point from two pointsets modeling faces. Dark blue lines indicate best matches given by $\mathcal{K}_{NS}$ while bright blue lines indicate best matches by the $\mathcal{K}_{NK}$. In both cases, $\mathcal{K}_{NK}$ gives the correct match while the $\mathcal{K}_{NS}$ fails. Computational complexity of $\mathcal{K}_{NK}$ is $O(k^2 n^2)$, $k$ being neighborhood size and $n$ number of points. The next section proposes a kernel which uses positions of points $(\mathbf{x}_i)$ in a neighborhood more strongly to calculate similarity in shape.

## 3.2 Spectral Neighborhood Kernel

The kernel defined in the previous section still uses a set of vectors kernel for finding similarity between the neighborhoods. Here, we are interested in a kernel function which evaluates the similarity in relative position of the corresponding points. Since the neighborhoods being compared are of fixed size, we assume that all points in a neighborhood have a corresponding point in the other. Thus, the correspondences are given by a permutation of points in one of the neighborhoods. This problem can be formulated as the weighted graph matching problem [16], for which spectral method is one of the popular heuristics. We use the features given by spectral decomposition of adjacency matrix of the neighborhood to define a kernel function.

Given a neighborhood $\mathcal{N}_i$ we define its adjacency matrix $\mathcal{A}_i$ as $\mathcal{A}_i(s,t) = e^{-\frac{\|\mathbf{x}_s - \mathbf{x}_t\|}{\alpha}}$, $\forall s,t | (\mathbf{x}_s, \mathbf{d}_s), (\mathbf{x}_t, \mathbf{d}_t) \in \mathcal{N}_i$, where $\alpha$ is a parameter. Given two neighborhoods $\mathcal{N}_i^A$ and $\mathcal{N}_j^B$, we are thus interested in a permutation $\pi$ of the basis of adjacency matrix of one of the neighborhoods (say $\mathcal{N}_j^B$), such that $\|\mathcal{A}_i^A - \pi(\mathcal{A}_j^B)\|_F$ is minimized, $\|.\|_F$ being the frobenius norm of a matrix.

It is well known that a matrix can be fully reconstructed from its spectral decomposition. Also, in the case that fewer eigenvectors are used, the equation $\|\mathcal{A} - \sum_{i=1}^{k} \lambda_i \zeta_i \zeta_i^T\|_F^2 = \sum_{j=k+1}^{n} \lambda_j^2$, suggests that eigenvectors corresponding to the higher eigenvalues will give better reconstruction. We use one eigenvector corresponding to largest eigenvalue. Thus, the approximate adjacency matrix becomes $\hat{\mathcal{A}} = \lambda_1 \zeta_1 \zeta_1^T$.

Let $\pi^*$ be the optimal permutation that minimizes $\|\hat{\mathcal{A}}_i^A - \pi(\hat{\mathcal{A}}_j^B)\|_F$. Note that here $\pi$ applied on a matrix implies permutation of the basis. It is easy to see that same permutation is induced on basis

of the eigenvectors $\zeta_j^B(1)$. Call $\mathbf{f}_i^A = |\zeta_i^A(1)|$ and $\mathbf{f}_j^B = |\zeta_j^B(1)|$, the *spectral projection* vectors corresponding to neighborhoods $\mathcal{N}_i^A$ and $\mathcal{N}_j^B$. Here $\zeta_i^A(1), \zeta_j^B(1)$ are eigenvectors corresponding to largest eigenvalue of $\hat{\mathcal{A}}_i^A, \hat{\mathcal{A}}_j^B$, and $|\zeta(1)|$ is the vector of absolute values of components of $\zeta(1)$. $f(s)$ can be thought of as projection of the $s^{th}$ point in the corresponding neighborhood on $\mathbb{R}^1$. It is equivalent to seek a permutation $\pi^*$ which minimizes $\|\mathbf{f}_i^A - \pi(\mathbf{f}_j^B)\|$, for comparing neighborhoods $\mathcal{N}_i^A$ and $\mathcal{N}_j^B$. The resulting similarity score is:

$$S(\mathcal{N}_i^A, \mathcal{N}_j^B) = \max_{\pi \in \Pi} T - \|\mathbf{f}_i^A - \pi(\mathbf{f}_j^B)\|_2^2 \tag{4}$$

where, $T$ is a threshold for converting the distance measure to similarity, and $\Pi$ is the set of all permutations. However, this similarity function is not necessarily positive semidefinite.

To construct a positive semidefinite kernel giving similarity between the vectors $\mathbf{f}_i^A$ and $\mathbf{f}_j^B$, we use the convolution kernel technique [7] on discrete structures. Let $x \in X$ be a composite object formed using parts from $X_1, \ldots, X_m$. Let $R$ be a relation over $X_1 \times \cdots \times X_m \times X$ such that $R(x_1, \ldots, x_m, x)$ is true if $x$ is composed of $x_1, \ldots, x_m$. Let $R^{-1}(x) = (x_1, \ldots, x_m) \in X_1 \times \cdots \times X_m | R(x_1, \ldots, x_m, x) = $ true and $\mathcal{K}^1, \ldots, \mathcal{K}^m$ be kernels on $X_1, \ldots, X_m$, respectively. The convolution kernel $K$ over $X$ is defined as:

$$\mathcal{K}(x, y) = \sum_{(x_1, \ldots, x_m) \in R^{-1}(x), (y_1, \ldots, y_m) \in R^{-1}(y)} \prod_{i=1}^{m} \mathcal{K}^i(x_i, y_i) \tag{5}$$

Haussler [7] showed that if $\mathcal{K}^1, \ldots, \mathcal{K}^m$ are symmetric and positive semidefinite, so is $\mathcal{K}$.

For us, let $X$ be the set of all neighborhoods and $X_1, \ldots, X_m$ be the sets of spectral projections of all points from all the neighborhoods. Here, note that even if the same point appears in different neighborhoods, the appearances will be considered to be different because the projections are relative to the neighborhoods. Since, each neighborhood has size $k$, in our case $m = k$. The relation $R$ is defined as $R(f(1), \ldots, f(k), \mathcal{N}_i^A)$ is true iff the vector $(f(1), \ldots, f(k)) = \pi(f_i^A)$ for some permutation $\pi$. In other words, $R(f(1), \ldots, f(k), \mathcal{N}_i^A)$ is true iff $f(1), \ldots, f(k)$ are spectral projections the points of neighborhood $\mathcal{N}_i^A$). Also, let $\mathcal{K}^i$, $i = 1 \ldots k$ all be RBF kernels with the same parameter $\beta$. Thus, from the above equation, the convolution kernel becomes $\mathcal{K}(N_i^A, N_j^B) = k! \sum_{\pi \in \Pi} e^{\frac{-1}{\beta} \sum_{l=1}^{l} (f_i^A(l) - f_j^B(\pi(l)))^2} = k! \sum_{\pi \in \Pi} e^{\frac{-\|\mathbf{f}_i^A - \pi(\mathbf{f}_j^B)\|^2}{\beta}}$. Dividing by the constant $(k!)^2$, we get kernel $\mathcal{K}_{SN}$ as:

$$\mathcal{K}_{SN}(\mathcal{N}_i^A, \mathcal{N}_j^B) = \frac{1}{k!} \sum_{\pi \in \Pi} e^{\frac{-\|\mathbf{f}_i^A - \pi(\mathbf{f}_j^B)\|^2}{\beta}} \tag{6}$$

The spectral kernel (SK) $\mathcal{K}_{SK}$ between two pointsets $X^A$ and $X^B$ is thus defined as:

$$\mathcal{K}_{SK}(X^A, X^B) = \frac{1}{n_A n_B} \sum_{i=1}^{n_A} \sum_{j=1}^{n_B} \mathcal{K}_{RBF}(\mathbf{d}_i^A, \mathbf{d}_j^B) \mathcal{K}_{SN}(\mathcal{N}_i^A, \mathcal{N}_j^B) \tag{7}$$

Following theorem relates $\mathcal{K}_{SN}(\mathcal{N}_i^A, \mathcal{N}_j^B)$ to $S(\mathcal{N}_i^A, \mathcal{N}_j^B)$ (eqn 4).

**Theorem 3.1** *Let $N_i$ and $N_j$ be two sub-structures with spectral projection vectors $f^i$ and $f^j$. For large enough value of $T$ such that all points are matched.*

$$\lim_{\beta \to 0} \mathcal{K}_{SN}(N_i, N_j))^\beta = \frac{e^{-T}}{k!} e^{S(N_i, N_j)}$$

**Proof:** Let $\pi^*$ be the permutation that gives the optimal score $S(N_i, N_j)$. By definition, $e^{S(N_i, N_j)} = e^T e^{-\|f^i - \pi^*(f^j)\|^2}$.

$$\lim_{\beta \to 0} (\mathcal{K}_{SN}(N_i, N_j))^\beta = \lim_{\beta \to 0} (\frac{1}{k!} \sum_{\pi \in \Pi(l)} e^{\frac{-\|f^i - \pi(f^j)\|^2}{\beta}})^\beta$$
$$= \frac{1}{k!} e^{-\|f^i - \pi^*(f^j)\|^2} \lim_{\beta \to 0} (1 + \sum_{\pi \in \Pi \setminus \{\pi^*\}} e^{\frac{-1}{\beta} (\|f^i - \pi(f^j)\|^2 - \|f^i - \pi^*(f^j)\|^2)})^\beta$$
$$= \frac{-1}{k!} e^{-\|f^i - \pi^*(f^j)\|^2}$$

Table 1: Recognition accuracy on AR face dataset (section 4.1)

| | Smile | Angry | Scream | Glasses | Scarf | Left-Light | Right-Light |
|---|---|---|---|---|---|---|---|
| 1-NN | 96.3% | 88.9% | 57.0% | 48.1% | 3.0% | 22.2% | 17.8% |
| PCA | 94.1% | 79.3% | 44.4% | 32.9% | 2.2% | 7.4% | 7.4% |
| LEM | 78.6% | 92.9% | 31.3% | 74.8% | 47.4% | 92.9% | 91.1% |
| AMM | 96.0% | 96.0% | 56.0% | 80.0% | 82.0% | NA | NA |
| Face-ARG | 97.8% | 96.3% | 66.7% | 80.7% | 85.2% | 98.5% | 96.3% |
| Sum(eq (1)) | 96.19% | 95.23% | 83.80% | 89.52% | 60.00% | 86.66% | 80.95% |
| NK (eq (3)) | 98.09% | 98.09% | 85.71% | 94.28% | 65.71% | 92.38% | 86.66% |
| SK (eq (7)) | 99.04% | 99.04% | 86.66% | 93.33% | 65.71% | 90.47% | 84.76% |

□

Computational complexity of this kernel is $O(k!n^2)$, where $k$ is neighborhood size and $n$ is no. of descriptor points. However, since in practice only small neighborhood sizes are considered, the computation time doesn't become prohibitive.

# 4 Experimental Results

In order to study the effectiveness of proposed kernels for practical visual tasks, we applied them on three problems. Firstly, the kernels were applied to the well known problem of face recognition [20], and results on two benchmark datasets (AR and ORL) were compared to existing state-of-the-art methods. Next we used the spectral kernel to tag images in personal photo albums using faces of people present in them. Finally, the spectral kernel was used for annotation of video sequences using faces of people present.

**Attribute** For face recognition, faces were modeled as attributed pointsets using local gabor descriptors [10] calculated at the corner points using Harris corner point detector [6]. At each point, gabor despite for three different scales and four different orientations were calculated. Descriptors for 5 points (4 pixel neighbors and itself) were used for each of the 12 combinations, making a total of 60 descriptors per point. For image tagging and video annotation, faces were modeled as attributed pointsets using SIFT local descriptors [11], having 128 descriptors per point.

The kernels were implemented in GNU C/C++. LAPACK [2] was used for calculation of eigenvectors and GNU GSL for calculation of permutations. LIBSVM [4] was used as the SVM based classifier for classifying pointsets. The face detector provided in OpenCV was used for detecting faces in album images and video frames.

**Dataset** The AR dataset [13] is composed of color images of 135 people (75 men and 60 women). The DB includes frontal view images with different facial expressions, illumination conditions, and occlusion by sunglasses and scarf. After removing persons with corrupted images or missing any of the 8 types of required images, a total of 105 persons (56 men and 49 women) were selected. All the images were converted to greyscale and rescaled to $154 \times 115$ pixels. The ORL dataset is composed of 10 images for each of the 40 persons. The images have minor variations in pose, illumination and scale. All the 400, $112 \times 92$ pixel images were used for experiments.

## 4.1 Face Recognition in AR face DB

The kernels proposed in this paper, were tested pointsets derived from images in AR face DB. Face recognition was posed as a multiclass classification problem, and SVMs were along with the proposed kernels. The AR face DB is a standard benchmark dataset, on which a recent comparison of state of the art methods for face recognition has been given in [14]. In table 1, we have restated the results provided in [14] along with the results of our kernels. All the results reported in table 1 have been obtained using one normal (no occlusion or change of expression) face image as the training set.

It can be seen that for all the images showing change of expression (Smile, Angry and Scream), the pointset kernels outperform existing methods. Also, in case of occlusion of face by glasses, the

Table 2: Recognition accuracy on ORL dataset (section 4.2)

| # of training images → | 1 | 3 | 5 |
|---|---|---|---|
| Sum (eq (1)) | 70.83% | 92.50% | 98.00% |
| NK (eq (3)) | 71.38% | 93.57% | 98.00% |
| SK (eq (7)) | 71.94% | 93.92% | 98.00% |

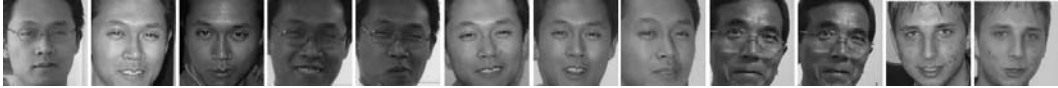

Figure 2: Representative cluster from tagging of album

pointset kernels give better results than existing methods. However, in case of occlusion by scarf, the kernel based method do not perform as well as the Face-ARG or AMM. This failure is due to introduction of a large number of points in the scarf themselves. It was observed that about 50% of the descriptor points in the faces having scarfs were in the scarf region of the image. Summing the similarities over such a large number of extra points makes the overall kernel value noisy.

The proposed approach doesn't perform better than existing methods on images taken under extreme variation in lighting conditions. This is due to the fact that values of the local descriptors change drastically with illumination. Also, some of the corner points disappear under different lighting condition. However, performance of the kernels is comparable to the existing methods, thus demonstrating the effectiveness of modeling faces as attributed pointsets.

## 4.2 Recognition performance on ORL Dataset

Real life problems in face recognition also show minor variations in pose, which are addressed by testing the kernels on images in the ORL dataset. The problem was posed as a multiclass classification problem and SVM was used along with the kernels for classification. Table 2 reports the recognition accuracies of all the three kernels for two different values of parameters, and for 1, 3 and 5 training images.

It can be seen that even with images showing minor variations in pose, the proposed kernels perform reasonably well. Also, due to change in pose the relative position of points in the pointsets change. This is reflected in the fact that improvement due to addition of position information in kernels is minor as compared to those shown in AR dataset. For higher number of training images, the performance of all the kernels saturate at 98%.

## 4.3 Tagging images in personal albums based on faces

The problem of tagging images in personal albums with names of people present in them, is a problem of high practical relevance [19]. The spectral kernels were used solve this problem. Images from publicly available sources like http://www.flickr.com [1] were used for experimentation. Five personal albums having 20 - 55 images each were downloaded and many images had upto 6 people. Face detector from openCV library was used to automatically detect faces in images. Detected faces are cropped and resized to $100 \times 100$ px resolution. 47 - 265 such faces detected from each album. To the best of our knowledge, there are no openly available techniques to benchmark our method against.

Due to non-availability of training data, the problem of image tagging was posed as a clustering problem. Faces detected from the images were represented as attributed pointsets using SIFT local descriptors, and spectral kernel was evaluated on them. A threshold based clustering scheme was used on the distance metric induced by the kernel ($d(x, y) = \sqrt{K(x, x) + K(y, y) - 2 * k(x, y)}$). Ideally, each cluster thus obtained should represent a person and images containing faces from a given cluster should be tagged with the name of that person.

Table 3: Face based album tagging

| Album no. | No. of people | | % Identified | % False +ve |
|---|---|---|---|---|
| | (Actual) | (Identified) | | |
| 1 | - | 2 | 90% | 0% |
| 2 | 14 | 6 | 84% | 10.52% |
| 3 | 8 | 4 | 66.66% | 8.33% |
| 4 | 4 | 2 | 83.33% | 19.44% |
| 5 | 3 | 2 | 80.00% | 14.70% |

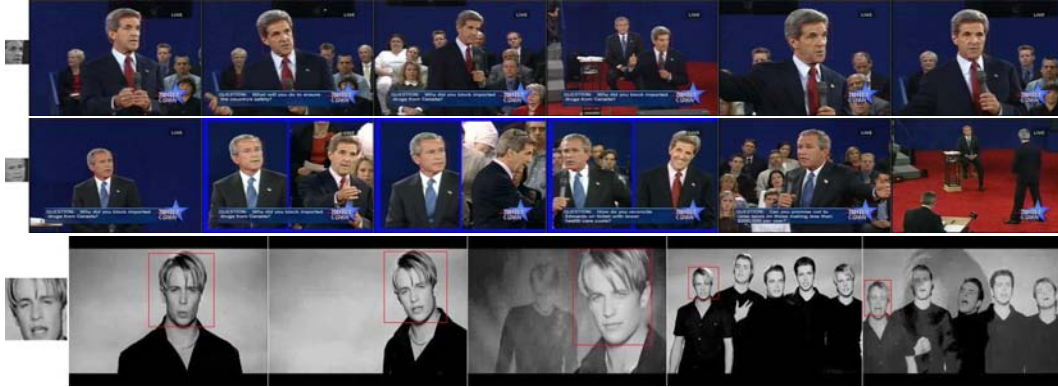

Figure 3: Keyframes of a few shots detected with annotation

Table 3 reports results from tagging experiments for five albums. No. of people identified reports the number clusters having more than one faces, as singleton cluster will always be correct for that person. Thus, people appearing only once in the entire album are not reported, which reduce the no. of identified people. % identified and % false +ve are averaged over all clusters detected in the album, and are calculated for each cluster as: $\% \ identified = \frac{No. \ of \ correct \ faces \ in \ the \ cluster}{Total \ no. \ of \ faces \ of \ the \ person}$ and $\% \ false + ve = \frac{false \ +ves \ in \ the \ cluster}{Total \ no. \ of \ faces \ in \ the \ cluster}$. It can be seen that the kernel performs reasonably well on the dataset. Figure 2 shows a representative cluster with the first 8 images as true +ves and rest as false +ves.

### 4.4 Video annotation based on faces

The kernels were also used to perform video shot annotation based faces detected in video sequences. Experimentation was performed on videos from "News and Public affair" section of www.archive.org and music videos from www.youtube.com. Video was sampled at 1 frame per second and experimental methodology was similar section 4.3 was used on the frames.

Figure 3 shows two representative shots from corresponding to two candidates from "Election 2004, presidential debate part 2", and one from "Westlife- Seasons in the Sun" video. The faces annotating the shots are shown in the left as thumbnails. It may be noted that for videos, high pose variation did not reduce accuracy of recognition due to gradual changing of pose. The results on detecting shots were highly encouraging, thus demonstrating the varied applicability of proposed attributed pointset kernels.

## 5 Conclusion

In this article, we propose kernels on attributed pointsets. We define the notion of neighborhood in an attributed pointset and propose two new kernels. The first kernel evaluates attribute similarities between the neighborhoods and uses the co-occurrence information to improve the performance of kernels on sets of vectors. The second kernel uses the position information more strongly and

matches the shapes of neighborhoods. This kernel function is motivated from spectral graph matching techniques.

The proposed kernels were validated on the well known task on face recognition on two popular benchmark datasets. Results show that the current kernels perform competitively with the state-of-the-art techniques for face recognition. The spectral kernel was also used to perform two real life tasks of tagging images in personal photo albums and annotating shots in videos. The results were encouraging in both cases.

## Footnotes

[1]Dept. of Computer Science & Automation, [2]Dept. of Electrical Engineering, Indian Institute of Science, Bangalore - 560012, India.

[1] We intend to make the dataset publicly available if no copyrights are violated

## References

[1] Helmut Alt and Leonidas J. Guibas. Discrete geometric shapes: Matching, interpolation, and approximation A survey. Technical Report B 96-11, 1996.

[2] E. Anderson, Z. Bai, C. Bischof, S. Blackford, J. Demmel, J. Dongarra, J. Du Croz, A. Greenbaum, S. Hammarling, A. McKenney, and D. Sorensen. *LAPACK Users' Guide*. Society for Industrial and Applied Mathematics, Philadelphia, PA, third edition, 1999.

[3] Karsten M. Borgwardt and Hans-Peter Kriegel. Shortest-path kernels on graphs. In *ICDM '05: Proceedings of the Fifth IEEE International Conference on Data Mining*, pages 74–81, Washington, DC, USA, 2005. IEEE Computer Society.

[4] Chih-Chung Chang and Chih-Jen Lin. *LIBSVM: a library for support vector machines*, 2001. Software available at http://www.csie.ntu.edu.tw/~cjlin/libsvm.

[5] Ingvar Eidhammer, Inge Jonassen, and William R. Taylor. Structure comparison and structure patterns. *Journal of Computational Biology*, 7(5):685–716, 2000.

[6] C. Harris and M.J. Stephens. A combined corner and edge detector. In *Proc. of Alvey Vision Conf.*, 1988.

[7] David Haussler. Convolution kernels on discrete structures. Technical report, University of California, Santa Cruz, 1999.

[8] Koji Tsuda Hisashi Kashima and Akihiro Inokuchi. Marginalized kernels between labeled graphs. In *Twentieth International Conference on Machine Learning (ICML)*, 2003.

[9] Risi Kondor and Tony Jebara. A kernel between sets of vectors. In *Twentieth International Conference on Machine Learning (ICML)*, 2003.

[10] Tai Sing Lee. Image representation using 2d gabor wavelets. *IEEE TPAMI*, 18(10):959–971, 1996.

[11] D. Lowe. Distinctive image features from scale-invariant keypoints. *Int. Journal of Computer Vision*, 20:91–110, 2003.

[12] Siwei Lyu. Mercer kernels for object recognition with local features. In *IEEE CVPR*, 2005.

[13] A.M. Martinez and R. Benavente. The ar face database. *CVC Technical Report*, 24, 1998.

[14] Bo Gun Park, Kyoung Mu Lee, and Sang Uk Lee. Face recognition using face-arg matching. *IEEE TPAMI*, 27(12):1982–1988, 2005.

[15] Amnon Shashua and Tamir Hazan. Algebraic set kernels with application to inference over local image representations. In *Neural Information Processing Systems (NIPS)*, 2004.

[16] Shinji Umeyama. An eigendecomposition approach to weighted graph matching problems. *IEEE transactions on pattern analysis and machine intelligence*, 10(5):695–703, 1988.

[17] Lior Wolf and Amnon Shashua. Learning over sets using kernel principal angles. *Journal of Machine Learning Research*, (4):913–931, 2003.

[18] Haim J. Wolfson and Isidore Rigoutsos. Geometric hashing: An overview. *IEEE Comput. Sci. Eng.*, 4(4):10–21, 1997.

[19] L. Zhang, L. Chen, M. Li, and H. Zhang. Automated annotation of human faces in family albums, 2003.

[20] W. Zhao, R. Chellappa, P. J. Phillips, and A. Rosenfeld. Face recognition: A literature survey. *ACM Comput. Surv.*, 35(4):399–458, 2003.

